# A Transductive Bound for the Voted Classifier with an Application to Semi-supervised Learning

**Massih R. Amini**
Laboratoire d'Informatique de Paris 6
Université Pierre et Marie Curie, Paris, France
massih-reza.amini@lip6.fr

**François Laviolette**
Université Laval
Québec (QC), Canada
francois.laviolette@ift.ulaval.ca

**Nicolas Usunier**
Laboratoire d'Informatique de Paris 6
Université Pierre et Marie Curie, Paris, France
nicolas.usunier@lip6.fr

## Abstract

We propose two transductive bounds on the risk of majority votes that are estimated over partially labeled training sets. The first one involves the margin distribution of the classifier and a risk bound on its associate Gibbs classifier. The bound is tight when so is the Gibbs's bound and when the errors of the majority vote classifier is concentrated on a zone of low margin. In semi-supervised learning, considering the margin as an indicator of confidence constitutes the working hypothesis of algorithms which search the decision boundary on low density regions. Following this assumption, we propose to bound the error probability of the voted classifier on the examples for whose margins are above a fixed threshold. As an application, we propose a self-learning algorithm which iteratively assigns pseudo-labels to the set of unlabeled training examples that have their margin above a threshold obtained from this bound. Empirical results on different datasets show the effectiveness of our approach compared to the same algorithm and the TSVM in which the threshold is fixed manually.

## 1 Introduction

Ensemble methods [5] return a weighted vote of baseline classifiers. It is well known that under the PAC-Bayes framework [9], one can obtain an estimation of the generalization error (also called *risk*) of such majority votes (referred as *Bayes classifier*). Unfortunately, those bounds are generally not tight, mainly because they are indirectly obtain via a bound on a randomized combination of the baseline classifiers (called the *Gibbs classifier*). Although the PAC-Bayes theorem gives tight risk bounds of Gibbs classifiers, the bounds of their associate Bayes classifiers come at a cost of worse risk (trivially a factor of 2, or under some margin assumption, a factor of $1+\epsilon$). In practice the Bayes risk is often smaller than the Gibbs risk.

In this paper we present a transductive bound over the Bayes risk. This bound is also based on the risk of the associated Gibbs classifier, but it takes as an additional information the exact knowledge of the margin distribution of unlabeled data. This bound is obtained by analytically solving a linear program. The intuitive idea here is that given the risk of the Gibbs classifier and the margin distribution, the risk of the majority vote classifier is maximized when all its errors are located on low margin examples. We show that our bound is tight when the associated Gibbs risk can accurately be estimated and when the Bayes classifier makes most of its errors on low margin examples.

The proof of this transductive bound makes use of the (joint) probability over an unlabeled data set that the majority vote classifier makes an error and the margin is above a given threshold. This second result naturally leads to consider the conditional probability that the majority vote classifier makes an error knowing that the margin is above a given threshold.

This conditional probability is related to the concept that the margin is an *indicator of confidence* which is recurrent in semi-supervised self-learning algorithms [3,6,10,11,12]. These methods first train a classifier on the labeled training examples. The classifier outputs serve then to assign pseudo-class labels to unlabeled data having margin above a given threshold. The supervised method is retrained using the initial labeled set and its previous predictions on unlabeled data as additional labeled examples. Practical algorithms almost fix the margin threshold manually.

In the second part of the paper, we propose to find this margin threshold by minimizing the bound on the conditional probability. Empirical results on different datasets show the effectiveness of our approach compared to TSVM [7] and the same algorithm but with a manually fixed threshold as in [11]

In the remainder of the paper, we present, in section 2, our transductive bounds and show their outcomes in terms of sufficient conditions under which unlabeled data may be of help in the learning process and a linear programming method to estimate these bounds. In section 4, we present experimental results obtained with a self-learning algorithm on different datasets in which we use the bound presented in section 2.2 for choosing the threshold which serve in the label assignment step of the algorithm. Finally, in section 5 we discuss the outcomes of this study and give some pointers to further research.

## 2 Transductive Bounds on the Risk of the Voted Classifier

We are interested in the study of binary classification problems where the input space $\mathcal{X}$ is a subset of $\mathbb{R}^d$ and the output space is $\mathcal{Y} = \{-1, +1\}$. We furthermore suppose that the training set is composed of a *labeled set* $Z_\ell = ((\mathbf{x}_i, y_i))_{i=1}^l \in \mathcal{Z}^l$ and an *unlabeled set* $X_{\mathcal{U}} = (\mathbf{x}_i')_{i=l+1}^{l+u} \in \mathcal{X}^u$, where $\mathcal{Z}$ represents the set of $\mathcal{X} \times \mathcal{Y}$. We suppose that each pair $(\mathbf{x}, y) \in Z_\ell$ is drawn i.i.d. with respect to a fixed, but unknown, probability distribution $\mathcal{D}$ over $\mathcal{X} \times \mathcal{Y}$ and we denote the marginal distribution over $\mathcal{X}$ by $\mathcal{D}_{\mathcal{X}}$.

To simplify the notation and the proofs, we restrict ourselves to the deterministic labeling case, that is, for each $\mathbf{x}' \in X_{\mathcal{U}}$, there is exactly one possible label that we will denote by $y'$.[1]

In this study, we consider learning algorithms that work in a fixed hypothesis space $\mathcal{H}$ of binary classifiers (defined without reference to the training data). After observing the training set $\mathcal{S} = Z_\ell \cup X_{\mathcal{U}}$, the task of the learner is to choose a *posterior* distribution $Q$ over $\mathcal{H}$ such that the $Q$-weighted majority vote classifier $B_Q$ (also called the *Bayes* classifier) will have the smallest possible risk on examples of $X_{\mathcal{U}}$. Recall that the Bayes classifier is defined by

$$B_Q(\mathbf{x}) = \text{sgn}\left[\mathbb{E}_{h \sim Q} h(\mathbf{x})\right] \qquad \forall \mathbf{x} \in \mathcal{X}. \tag{1}$$

where, sgn($x$)=+1 if the real number $x > 0$ and $-1$ otherwise. We further denote by $G_Q$ the associated *Gibbs* classifier which for classifying any example $\mathbf{x} \in \mathcal{X}$ chooses randomly a classifier $h$ according to the distribution $Q$. We accordingly define the *transductive* risk of $G_Q$ over an unlabeled set by:

$$R_u(G_Q) \stackrel{def}{=} \frac{1}{u} \sum_{\mathbf{x}' \in X_{\mathcal{U}}} \mathbb{E}_{h \sim Q} [\![ h(\mathbf{x}') \neq y' ]\!] \tag{2}$$

Where, $[\![ \pi ]\!] = 1$ if predicate $\pi$ holds and 0 otherwise, and for every unlabeled example $\mathbf{x}' \in X_{\mathcal{U}}$ we refer to $y'$ as its true *unknown* class label. In section 2.1 we show that if we consider the margin as an indicator of confidence and that we dispose a tight upper bound $R_u^\delta(G_Q)$ of the risk of $G_Q$ which holds with probability $1 - \delta$ over the random choice of $Z_\ell$ and $X_{\mathcal{U}}$ (for example using Theorem 17 or 18 of Derbelo *et al.* [4]), we are then able to accurately bound the *transductive* risk of the Bayes classifier:

$$R_u(B_Q) \stackrel{def}{=} \frac{1}{u} \sum_{\mathbf{x}' \in X_{\mathcal{U}}} [\![ B_Q(\mathbf{x}') \neq y' ]\!] \tag{3}$$

This result follows from a bound on the joint Bayes risk:

$$R_{u \wedge \theta}(B_Q) \stackrel{def}{=} \frac{1}{u} \sum_{\mathbf{x}' \in X_{\mathcal{U}}} [\![ B_Q(\mathbf{x}') \neq y' \wedge m_Q(\mathbf{x}') > \theta ]\!] \tag{4}$$

Where $m_Q(\cdot) = |\mathbb{E}_{h \sim Q} h(\cdot)|$ denotes the unsigned margin function. One of the practical issues that arises from this result is the possibility to define a threshold $\theta$ for which the bound is optimal and that we use in a self-learning algorithm by iteratively assigning pseudo-labels to unlabeled examples having margin above this threshold. We finally denote by $\mathbb{E}_u z$ the expectation of a random variable $z$ with respect to the *uniform distribution* over $X_{\mathcal{U}}$ and for notation convenience we equivalently define $P_u$ the uniform probability distribution over $X_{\mathcal{U}}$ i.e. For any subset $A$, $P(A) = \frac{1}{u} card(A)$.

## 2.1 Main Result

Our main result is the following theorem which provides two bounds on the transductive risks of the Bayes classifier (3) and the joint Bayes risk (4).

**Theorem 1** *Suppose that $B_Q$ is as in (1). Then for all $Q$ and all $\delta \in (0,1]$ with probability at least $1 - \delta$:*

$$R_u(B_Q) \leq \inf_{\gamma \in (0,1]} \left\{ P_u(m_Q(\mathbf{x}') < \gamma) + \frac{1}{\gamma} \left\lfloor K_u^\delta(Q) - M_Q^<(\gamma) \right\rfloor_+ \right\} \tag{5}$$

*Where $K_u^\delta(Q) = R_u^\delta(G_Q) + \frac{1}{2}\left( \mathrm{E}_u m_Q(\mathbf{x}') - 1\right)$, $M_Q^\lhd(t) = \mathrm{E}_u m_Q(\mathbf{x}') [\![m_Q(\mathbf{x}') \lhd t]\!]$ for $\lhd$ being $<$ or $\leq$ and $\lfloor . \rfloor_+$ denotes the positive part (i.e. $\lfloor x \rfloor_+ = [\![x > 0]\!]x$).*

*More generally, with probability at least $1 - \delta$, for all $Q$ and all $\theta \geq 0$:*

$$R_{u \wedge \theta}(B_Q) \leq \inf_{\gamma \in (\theta, 1]} \left\{ P_u(\theta < m_Q(\mathbf{x}') < \gamma) + \frac{1}{\gamma} \left\lfloor K_u^\delta(Q) + M_Q^\leq(\theta) - M_Q^<(\gamma) \right\rfloor_+ \right\} \tag{6}$$

In section 2.2 we will prove that the bound (5) simply follows from (6). In order to better understand the former bound on the risk of the Bayes classifier, denote by $F_u^\delta(Q)$ the right hand side of equation (5):

$$F_u^\delta(Q) \stackrel{def}{=} \inf_{\gamma \in (0,1]} \left\{ P_u(m_Q(\mathbf{x}') < \gamma) + \frac{1}{\gamma} \left\lfloor K_u^\delta(Q) - M_Q^<(\gamma) \right\rfloor_+ \right\}$$

and consider the following special case where the classifier makes *most* of its errors on unlabeled examples with low margin. Proposition 2, together with the explanations that follow, makes this idea clearer.

**Proposition 2** *Assume that $\forall \mathbf{x} \in X_{\mathcal{U}}, m_Q(\mathbf{x}) > 0$ and that $\exists C \in (0,1]$ such that $\forall \gamma > 0$:*

$$P_u\left( B_Q(\mathbf{x}') \neq y' \wedge m_Q(\mathbf{x}') = \gamma \right) \neq 0 \Rightarrow P_u\left( B_Q(\mathbf{x}') \neq y' \wedge m_Q(\mathbf{x}') < \gamma \right) \geq C \cdot P_u\left( m_Q(\mathbf{x}') < \gamma \right)$$

*Then, with probability at least $1 - \delta$:*

$$F_u^\delta(Q) - R_u(B_Q) \leq \frac{1 - C}{C} R_u(B_Q) + \frac{R_u^\delta(G_Q) - R_u(G_Q)}{\gamma^*} \tag{7}$$

*Where $\gamma^* = \sup \left\{ \gamma | P_u\left( B_Q(\mathbf{x}') \neq y' \wedge m_Q(\mathbf{x}') = \gamma \right) \neq 0 \right\}$*

Now, suppose that the margin is an indicator of confidence. Then, a Bayes classifier that makes its error mostly on low margin regions will admit a coefficient $C$ in inequality (7) close to 1 and the bound of (5) becomes tight (provided we have an accurate upper bound $R_u^\delta(G_Q)$ ). In the next section we provide proofs of all the statements above and show in lemma 4 a simple way to compute the best margin threshold for which the general bound on the joint Bayes risk is the lowest.

## 2.2 Proofs

All our proofs are based on the relationship between $R_u(G_Q)$ and $R_u(B_Q)$ and the following lemma:

**Lemma 3** *Let $(\gamma_1, .., \gamma_N)$ be the ordered sequence of the different strictly positive values of the margin on $X_{\mathcal{U}}$, that is $\{\gamma_i, i = 1..N\} = \{m_Q(\mathbf{x}') | \mathbf{x}' \in X_{\mathcal{U}} \wedge m_Q(\mathbf{x}') > 0\}$ and $\forall i \in \{1, \ldots, N-1\}, \gamma_i < \gamma_{i+1}$. Denote moreover $b_i = P_u\left( B_Q(\mathbf{x}') \neq y' \wedge m_Q(\mathbf{x}') = \gamma_i \right)$ for $i \in \{1, \ldots, N\}$. Then,*

$$R_u(G_Q) = \sum_{i=1}^{N} b_i \gamma_i + \frac{1}{2}\left( 1 - \mathrm{E}_u m_Q(\mathbf{x}') \right) \tag{8}$$

$$\forall \theta \in [0,1], R_{u \wedge \theta}(B_Q) = \sum_{i=k+1}^{N} b_i \quad \text{with } k = \max\{i | \gamma_i \leq \theta\} \tag{9}$$

**Proof** Equation (9) follows the definition $R_{u \wedge \theta}(B_Q) = P_u\left( B_Q(\mathbf{x}') \neq y' \wedge m_Q(\mathbf{x}') > \theta \right)$.

Equation (8) is obtained from the definition of the margin $m_Q$ which writes as

$$\forall \mathbf{x}' \in X_{\mathcal{U}}, m_Q(\mathbf{x}') = |\mathrm{E}_{h \sim Q}[\![h(\mathbf{x}') = 1]\!] - \mathrm{E}_{h \sim Q}[\![h(\mathbf{x}') = -1]\!]| = |1 - 2\mathrm{E}_{h \sim Q}[\![h(\mathbf{x}') \neq y']\!]|$$

By noticing that for all $\mathbf{x}' \in X_{\mathcal{U}}$ the condition $\mathrm{E}_{h\sim Q}[\![h(\mathbf{x}') \neq y']\!] > \frac{1}{2}$ is equivalent to the statement $y'\mathrm{E}_{h\sim Q}h(\mathbf{x}') < 0$ or $B_Q(\mathbf{x}') \neq y'$, we can rewrite $m_Q$ without absolute values and hence get:

$$\mathrm{E}_{h\sim Q}[\![h(\mathbf{x}') \neq y']\!] = \frac{1}{2}(1 + m_Q(\mathbf{x}'))[\![B_Q(\mathbf{x}') \neq y']\!] + \frac{1}{2}(1 - m_Q(\mathbf{x}'))[\![B_Q(\mathbf{x}') = y']\!] \qquad (10)$$

Finally equation (8) yields by taking the mean over $\mathbf{x}' \in X_{\mathcal{U}}$ and by reorganizing the equation using the notations of $b_i$ and $\gamma_i$. Recall that the values the $\mathbf{x}'$ for which $m_Q(\mathbf{x}') = 0$ counts for 0 in the sum that defined the Gibbs risk (see equation 2 and the definition of $m_Q$). $\square$

**Proof of Theorem 1** First, we notice that equation (5) follows equation (6) from the fact that $M_Q^{\leq}(0) = 0$ and the following inequality:

$$R_u(B_Q) = R_{u \wedge 0}(B_Q) + P_u(B_Q(\mathbf{x}') \neq y' \wedge m_Q(\mathbf{x}') = 0) \leq R_{u \wedge 0}(B_Q) + P_u(m_Q(\mathbf{x}') = 0)$$

For proving equation (6), we know from lemma 3 that for a fix $\theta \in [0,1]$ there exist $(b_1, \ldots, b_N)$ such that $0 \leq b_i \leq P_u(m_Q(\mathbf{x}') = \gamma_i)$ and which satisfy equations (8) and (9).

Let $k = \max\{i \mid \gamma_i \leq \theta\}$, assuming now that we can obtain an upper bound $R_u^\delta(G_Q)$ of $R_u(G_Q)$ which holds with probability $1 - \delta$ over the random choices of $Z_\ell$ and $X_{\mathcal{U}}$, from the definition (4) of $R_{u \wedge \theta}(B_Q)$ with probability $1 - \delta$ we have then

$$R_{u \wedge \theta}(B_Q) \leq \max_{b_1,..,b_N} \sum_{i=k+1}^{N} b_i \text{ u.c. } \forall i, 0 \leq b_i \leq P_u(m_Q(\mathbf{x}') = \gamma_i) \text{ and } \sum_{i=1}^{N} b_i \gamma_i \leq K_u^\delta(Q) \qquad (11)$$

Where $K_u^\delta(Q) = R_u^\delta(G_Q) - \frac{1}{2}(1 - \mathrm{E}_u m_Q(\mathbf{x}'))$. It turns out that the right hand side in equation (11) is the solution of a linear program that can be solved analytically and which is attained for:

$$b_i = \begin{cases} 0 & \text{if } i \leq k, \\ \min\left(P_u(m_Q(\mathbf{x}') = \gamma_i), \left\lfloor \frac{K_u^\delta(Q) - \sum_{k<j<i} \gamma_j P_u(m_Q(\mathbf{x}')=\gamma_j)}{\gamma_i} \right\rfloor_+\right) & \text{elsewhere.} \end{cases} \qquad (12)$$

For clarity, we defer the proof of equation (12) to lemma 4, and continue the proof of equation (6).

Using the notations defined in Theorem 1, we rewrite $\sum_{k<j<i} \gamma_j P_u(m_Q(\mathbf{x}') = \gamma_j)$ as $M_Q^{\leq}(\gamma_i) - M_Q^{\leq}(\theta)$. We further define $I = \max\left\{i | K_u^\delta(Q) + M_Q^{\leq}(\theta) - M_Q^{\leq}(\gamma_i) > 0\right\}$ which implies $\sum_{i=k+1}^{N} b_i = \sum_{i=k+1}^{I} b_i$ from equations (11) and (12) with $b_I = \frac{K_u^\delta(Q) + M_Q^{\leq}(\theta) - M_Q^{\leq}(\gamma_I)}{\gamma_I}$. From this inequality we hence obtain a bound on $R_{u \wedge \theta}(B_Q)$:

$$R_{u \wedge \theta}(B_Q) \leq P_u(\theta < m_Q(\mathbf{x}') < \gamma_I) + \frac{K_u^\delta(Q) + M_Q^{\leq}(\theta) - M_Q^{\leq}(\gamma_I)}{\gamma_I} \qquad (13)$$

The proof of the second point in theorem 1 is just a rewriting of this result as from the definition of $\gamma_I$, for any $\gamma > \gamma_I$, the right-hand side of equation (6) is equal to $P_u(m_Q(\mathbf{x}') < \gamma)$, which is greater than the right-hand side of equation (13). Moreover, for $\gamma < \gamma_I$, we notice that $\gamma \mapsto P_u(m_Q(\mathbf{x}') < \gamma) + \frac{K_u^\delta(Q) + M_Q^{\leq}(\theta) - M_Q^{\leq}(\gamma)}{\gamma}$ decreases. $\square$

**Lemma 4 (equation** (12)) *Let* $g_i, i = 1...N$ *be such that* $0 < g_i < g_{i+1}$, $p_i \geq 0, i = 1...N$, $B \geq 0$ *and* $k \in \{1, \ldots, N\}$. *Then, the optimal value of the linear program:*

$$\max_{q_1,...,q_N} \sum_{i=k+1}^{N} q_i \quad \text{u.c. } \forall i, 0 \leq q_i \leq p_i \text{ and } \sum_{i=1}^{N} q_i g_i \leq B \qquad (14)$$

*is attained for* $q^*$ *defined by:* $\forall i \leq k : q_i^* = 0$ *and* $\forall i > k, q_i^* = \min\left(p_i, \lfloor \frac{B - \sum_{j<i} q_j^* g_j}{g_i} \rfloor_+\right)$

**Proof** Define $\mathcal{O} = \{0\}^k \times \prod_{i=k+1}^{N} [0, p_i]$. We will show that problem (14) has a unique optimal solution in $\mathcal{O}$, and that this solution is $q^*$. In the rest of the proof, we denote $F(q) = \sum_{i=k+1}^{N} q_i$.

First, the problem is convex, feasible (take $\forall i, q_i = 0$) and bounded. Therefore there is an optimal solution $q^{opt} \in \prod_{i=1}^{N}[0, p_i]$. Define $q^{opt,\mathcal{O}}$ by $q_i^{opt,\mathcal{O}} = q_i^{opt}$ if $i > k$ and $q_i^{opt,\mathcal{O}} = 0$ otherwise. Then, $q^{opt,\mathcal{O}} \in \mathcal{O}$, it is clearly feasible, and $F(q^{opt,\mathcal{O}}) = F(q^{opt})$. Therefore, there is an optimal solution in $\mathcal{O}$.

Now, for $(q, q') \in \mathbb{R}^N \times \mathbb{R}^N$, define $\mathbb{I}(q, q') = \{i | q_i > q_i'\}$, and consider the lexicographic order $\succeq$:

$$\forall (q, q') \in \mathbb{R}^N \times \mathbb{R}^N, q \succeq q' \Leftrightarrow \mathbb{I}(q', q) = \emptyset \text{ or } (\mathbb{I}(q, q') \neq \emptyset \text{ and } \min \mathbb{I}(q, q') < \min \mathbb{I}(q', q))$$

The crucial point is that $q^*$ is the greatest feasible solution in $\mathcal{O}$ for $\succeq$. Indeed, notice first that $q^*$ is necessarily feasible and $\sum_{i=1}^{N} q_i^* = B$. To see this result let $\mathcal{M}$ be the set $\{i > k| : q_i^* < p_i\}$, we then have two possibilities to consider. *(a)* $\mathcal{M} = \emptyset$. In this case $q^*$ is simply the maximal element for $\succeq$ in $\mathcal{O}$. *(b)* $\mathcal{M} \neq \emptyset$. In this case, let $K = \min\{i > k | q_i^* < p_i\}$ and $M = \mathbb{I}(q, q^*)$.

We claim that there are no feasible $q \in \mathbb{R}^N$ such that $q \succ q^*$. By way of contradiction, suppose such a $q$ exists. Then, if $M \leq k$, we have $q_M > 0$, and therefore $q$ is not in $\mathcal{O}$; if $k < M < K$, we have $q_M > p_M$, which yields the same result; and finally, if $M \geq K$, we have $\sum_{i=1}^{N} q_i > \sum_{i=1}^{N} q_i^* = B$, and $q$ is not feasible.

We now show that if $q \in \mathcal{O}$ is feasible and $q^* \succ q$, then $q$ is not optimal (which is equivalent to show that *an optimal solution in $\mathcal{O}$ must be the greatest feasible solution for $\succeq$*).

Let $q \in \mathcal{O}$ be a feasible solution such that $q^* \succ q$. Since $q \succ q^*$, $\mathbb{I}(q^*, q)$ is not empty. If $\mathbb{I}(q, q^*) = \emptyset$, we have $F(q^*) > F(q)$, and therefore $q$ is not optimal. We now treat the case where $\mathbb{I}(q, q^*) \neq \emptyset$.

Let $K = \min \mathbb{I}(q^*, q)$ and $M = \min \mathbb{I}(q, q^*)$. We have $q_M > 0$ by definition, and $K < M$ because $q^* \succ q$ and $q \in \mathcal{O}$. Let $\lambda = \min\left(q_M, \frac{g_M}{g_K}(p_K - q_K)\right)$ and define $q'$ by:

$$q_i' = q_i \text{ if } i \notin \{K, M\}, \qquad q_K' = q_K + \frac{g_M}{g_K}\lambda \quad \text{and} \quad q_M' = q_M - \lambda.$$

We can see that $q'$ is feasible by the definition of $\lambda$, that it satisfies the box constraints, and $\sum_i q_i' g_i = \sum_i q_i g_i + \frac{g_M}{g_K}\lambda * g_K - \lambda * g_M = \sum_i q_i g_i \leq B$. Moreover $F(q') = F(q) + \lambda(\frac{g_M}{g_K} - 1) > F(q)$ since $g_K < g_M$ and $\lambda > 0$. Thus, $q$ is not optimal.

In summary, we have shown that there is an optimal solution in $\mathcal{O}$, and that a feasible solution in $\mathcal{O}$ must be the greatest feasible solution for the lexicographic order in $\mathcal{O}$ to be optimal and which is $q^*$. $\square$

**Proof of Proposition 2** First let us claim that

$$R_u(B_Q) \geq P_u\left(B_Q(\mathbf{x}') \neq y' \wedge m_Q(\mathbf{x}') < \gamma^*\right) + \frac{1}{\gamma^*}\left\lfloor K_u(Q) - M_Q^<(\gamma^*)\right\rfloor_+ \tag{15}$$

where $\gamma^* = \sup\left\{\gamma | P_u\left(B_Q(\mathbf{x}') \neq y' \wedge m_Q(\mathbf{x}') = \gamma\right) \neq 0\right\}$ and $K_u(Q) = R_u(G_Q) + \frac{1}{2}\left(\mathrm{E}_u m_Q(\mathbf{x}') - 1\right)$.

Indeed, assume for now that equation (15) is true. Then, by assumption we have:

$$R_u(B_Q) \geq C \cdot P_u\left(m_Q(\mathbf{x}') < \gamma^*\right) + \frac{1}{\gamma^*}\left\lfloor K_u(Q) - M_Q^<(\gamma^*)\right\rfloor_+ \tag{16}$$

Since $F_u^\delta(Q) \leq P_u\left(m_Q(\mathbf{x}') < \gamma^*\right) + \frac{1}{\gamma^*}\left\lfloor K_u^\delta(Q) - M_Q^<(\gamma^*)\right\rfloor_+$, with probability at least $1 - \delta$ we obtain:

$$F_u^\delta(Q) - R_u(B_Q) \leq (1 - C)P_u\left(m_Q(\mathbf{x}') < \gamma^*\right) + \frac{R_u^\delta(G_Q) - R_u(G_Q)}{\gamma^*} \tag{17}$$

This is due to the fact that $\left\lfloor K_u^\delta(Q) - M_Q^<(\gamma^*)\right\rfloor_+ - \left\lfloor K_u(Q) - M_Q^<(\gamma^*)\right\rfloor_+ \leq R_u^\delta(G_Q) - R_u(G_Q)$ when $R_u^\delta(G_Q) \geq R_u(G_Q)$. Taking once again equation (16), we have $P_u\left(m_Q(\mathbf{x}') < \gamma^*\right) \leq \frac{1}{C}R_u(B_Q)$. Plugging back this result in equation (17) yields Proposition 2.

Now, let us prove the claim (equation (15)). Since $\forall \mathbf{x}' \in X_\mathcal{U}, m_Q(\mathbf{x}') > 0$, we have $R_u(B_Q) = R_{u \wedge 0}(B_Q)$. Using the notations of lemma 3, denote $K$ the index such that $\gamma_K = \gamma^*$. Then, it follows from equation (8) of lemma 3 that $R_u(G_Q) = \sum_{i=1}^{K} b_i \gamma_i + \frac{1}{2}\left(1 - \mathrm{E}_u m_Q(\mathbf{x}')\right)$. Solving for $b_K$ in this equality yields $b_K = \frac{K_u(Q) - \sum_{i=1}^{K-1} b_i \gamma_i}{\gamma_K}$ and we therefore have $b_K \geq \frac{1}{\gamma_K}\lfloor K_u(Q) - M_Q^<(\gamma^*)\rfloor_+$ since $b_K \geq 0$ and $\forall i, b_i \leq P_u\left(m_Q(\mathbf{x}') = \gamma_i\right)$. Finally, from equation (9), we have $R_u(B_Q) = \sum_{i=1}^{K} b_i$, which implies equation (15) by using the lower bound on $b_K$ and the fact that $\sum_{i=1}^{K-1} b_i = P_u\left(B_Q(\mathbf{x}') \neq y' \wedge m_Q(\mathbf{x}') < \gamma^*\right)$. $\square$

In general, good PAC-Bayesian approximations of $R_u(G_Q)$ are difficult to carry out in supervised learning [4] mostly due to the huge number of needed instances to obtain accurate approximations of the distribution of the absolute values of the margin. In this section we have shown that the transductive setting allows for high precision on the bounds from the risk $R_u(G_Q)$ of the Gibbs classifier to the risk $R_u(B_Q)$ if we suppose that the Bayes classifier makes its errors mostly on low margin regions.

## 3 Relationship with margin-based self-learning algorithms

In Proposition 2 we have considered the hypothesis that *the margin is an indicator of confidence* as one of the sufficient conditions which leads to a tight approximation of the risk of the Bayes classifier, $R_u(B_Q)$. This assumption constitutes the working hypothesis of margin-based self-learning algorithms in which a classifier is first built on the labeled training set. The output of the learner can then be used to assign pseudo-labels to unlabeled examples having a margin above a fixed threshold (denoted by the set $Z_{\mathcal{U}}$ in what follows) and the supervised method is repeatedly retrained upon the set of the initial labeled and unlabeled examples that have been classified in the previous steps. The idea behind this pseudo-labeling is that unlabeled examples having a margin above a threshold are less subject to error prone labels, or equivalently, are those which have a small conditional Bayes error defined as:

$$R_{u|\theta}(B_Q) \overset{def}{=} P_u(B_Q(\mathbf{x}') \neq y' \mid m_Q(\mathbf{x}') > \theta) = \frac{R_{u \wedge \theta}(B_Q)}{P_u(m_Q(\mathbf{x}') > \theta)} \qquad (18)$$

In this case the label assignation of unlabeled examples upon a margin criterion has the effect to push away the decision boundary from the unlabeled data. This strategy follows the *cluster assumption* [10] used in the design of some semi-supervised learning algorithms where the decision boundary is supposed to pass through a region of low pattern density. Though margin-based self-learning algorithms are inductive in essence, their learning phase is nearly related to transductive learning which predicts the labels of a given unlabeled set. Indeed, in both cases the pseudo class-label assignation of unlabeled examples is interrelated to their margin.

For all these algorithms the choice of the threshold is a crucial point, as with a low threshold the risk to assign false labels to examples is high and a higher value of the threshold would not provide enough examples to enhance the current decision function. In order to examine the effect of fixing the threshold or computing it automatically we considered the margin-based self-training algorithm proposed by Tür *et al.* [10, Figure 6] (referred as SLA in the following), in which unlabeled examples having margin above a fixed threshold are iteratively added to the labeled

---
**Input:** Labeled and Unlabeled training sets: $Z_\ell$, $X_{\mathcal{U}}$
**Initialize**
(1) Train a classifier $H$ on $Z_\ell$
(2) Set $Z_{\mathcal{U}} \leftarrow \emptyset$
**repeat**
    (3) Compute the margin threshold $\theta^*$ minimizing (18) from (6)
    (4) $S \leftarrow \{(\mathbf{x}', y') \mid \mathbf{x}' \in X_{\mathcal{U}}; m_Q(\mathbf{x}') \geq \theta^* \wedge y' = sgn(H(\mathbf{x}'))\}$
    (5) $Z_{\mathcal{U}} \leftarrow Z_{\mathcal{U}} \cup S, X_{\mathcal{U}} = X_{\mathcal{U}} \backslash S$
    (6) Learn a classifier $H$ by optimizing a global loss function on $Z_\ell$ and $Z_{\mathcal{U}}$
**until** *$X_{\mathcal{U}}$ is empty or that there are no adds to $Z_{\mathcal{U}}$* ;
**Output** The final classifier $H$

Figure 1: Self-learning algorithm (SLA$^*$)

---

set and are not considered in next rounds for label distribution. In our approach, the best threshold minimizing the conditional Bayes error (18) from equation (6) of theorem 1 is computed at each round of the algorithm (line 3, figure 1 - SLA$^*$) while the threshold is kept fixed in [10, Figure 6] (line 3 is outside of the *repeat* loop). The bound $R_Q^\delta(G)$, of the risk of the Gibbs classifier which is involved in the computation of the threshold in equation (18) was fixed to its worst value 0.5.

## 4 Experiments and Results

In our experiments, we employed a Boosting algorithm optimizing the following exponential loss[2] as the baseline learner (line (6), figure 1):

$$\mathcal{L}_c(H, Z_\ell, Z_{\mathcal{U}}) = \frac{1}{l} \sum_{\mathbf{x} \in Z_\ell} e^{-yH(\mathbf{x})} + \frac{1}{|Z_{\mathcal{U}}|} \sum_{\mathbf{x}' \in Z_{\mathcal{U}}} e^{-y'H(\mathbf{x}')} \qquad (19)$$

Where $H = \sum_t \alpha_t h_t$ is a linear weighted sum of decision stumps $h_t$ which are uniquely defined by an input feature $j_t \in \{1, \ldots, d\}$ and a threshold $\lambda_t$ as:

$$h_t(\mathbf{x}) = 2[\![\varphi_{j_t}(\mathbf{x}) > \lambda_t]\!] - 1$$

With $\varphi_j(\mathbf{x})$ the $j^{th}$ feature characteristic of $\mathbf{x}$. Within this setting, the Gibbs classifier is defined as a random choice from the set of baseline classifiers $\{h_t\}_{t=1}^T$ according to $Q$ such that $\forall t, P_Q(h_t) = \frac{|\alpha_t|}{\sum_t |\alpha_t|}$. Accordingly the Bayes classifier is simply the weighted voting classifier $B_Q = sign(H)$. Although the self-learning model (SLA*) is an inductive algorithm we carried out experiments in a transductive setting in order to compare results with the transductive SVM of Joachims [7] and the self-learning algorithm (SLA) described in [11, Figure 6]. For the latter, after training a classifier $H$ on $Z_\ell$ (figure 1, step 1) we fixed different margin thresholds considering the lowest and the highest output values of $H$ over the labeled training examples. We evaluated the performance of the algorithms on 4 collections from the benchmark data sets[3] used in [3] as well as 2 data sets from the UCI repository [2]. In this case, we chose sets large enough for reasonable labeled/unlabeled partitioning, and that represent binary classification problems. Each experiment was repeated 20 times by partitioning, at each time, the data set into two random labeled and unlabeled training sets.

Table 1: Means and standard deviations of the classification error on unlabeled training data over the 20 trials for each data set. $d$ denotes the dimension, $l$ and $u$ refer respectively to the number of labeled and unlabeled examples in each data set.

| Dataset | $d$ | $l+u$ | $l$ | SLA | SLA* | TSVM | $l$ | SLA | SLA* | TSVM |
|---------|-----|-------|-----|-----|------|------|-----|-----|------|------|
| COIL$_2$ | 241 | 1500 | 10 | $.302^\downarrow_{\pm.042}$ | $\mathbf{.255}_{\pm.019}$ | $.286^\downarrow_{\pm.031}$ | 100 | $.148^\downarrow_{\pm.015}$ | $\mathbf{.134}_{\pm.011}$ | $.152^\downarrow_{\pm.016}$ |
| DIGIT | 241 | 1500 | 10 | $.201^\downarrow_{\pm.038}$ | $\mathbf{.149}_{\pm.012}$ | $.156_{\pm.014}$ | 100 | $.091^\downarrow_{\pm.01}$ | $\mathbf{.071}_{\pm.005}$ | $.087^\downarrow_{\pm.009}$ |
| G241c | 241 | 1500 | 10 | $.314^\downarrow_{\pm.037}$ | $\mathbf{.248}_{\pm.018}$ | $.252_{\pm.021}$ | 100 | $.201^\downarrow_{\pm.017}$ | $\mathbf{.191}_{\pm.014}$ | $.196_{\pm.022}$ |
| USPS | 241 | 1500 | 10 | $.342^\downarrow_{\pm.024}$ | $.278^\downarrow_{\pm.022}$ | $\mathbf{.261}_{\pm.019}$ | 100 | $.114^\downarrow_{\pm.012}$ | $.112^\downarrow_{\pm.012}$ | $\mathbf{.103}_{\pm.011}$ |
| PIMA | 8 | 768 | 10 | $.379^\downarrow_{\pm.026}$ | $\mathbf{.305}_{\pm.021}$ | $.318^\downarrow_{\pm.018}$ | 50 | $.284^\downarrow_{\pm.019}$ | $\mathbf{.266}_{\pm.018}$ | $.276_{\pm.021}$ |
| WDBC | 30 | 569 | 10 | $.168^\downarrow_{\pm.016}$ | $\mathbf{.124}_{\pm.011}$ | $.141^\downarrow_{\pm.016}$ | 50 | $.112^\downarrow_{\pm.011}$ | $\mathbf{.079}_{\pm.007}$ | $.108^\downarrow_{\pm.01}$ |

For each data set, means and standard deviations of the classification error on unlabeled training data over the 20 trials are shown in Table 1 for 2 different splits of the labeled and unlabeled sets. The symbol $\downarrow$ indicates that performance is significantly worse than the best result, according to a Wilcoxon rank sum test used at a p-value threshold of 0.01 [8]. In addition, we show in figure 2 the evolutions on the COIL$_2$, DIGIT and USPS data sets of the classification and both risks of the Gibbs classifier (on the labeled and unlabeled training sets) for different number of rounds in the SLA* algorithm. These figures are obtained from one of the 20 trials that we ran for these collections.

The most important conclusion from these empirical results is that for all data sets, the self-learning algorithm becomes competitive when the margin threshold is found automatically rather than if it is fixed manually. The *augmented* self-learning algorithm achieves performance statistically better or equivalent to that of TSVM in most cases, while it outperforms the initial method over all runs.

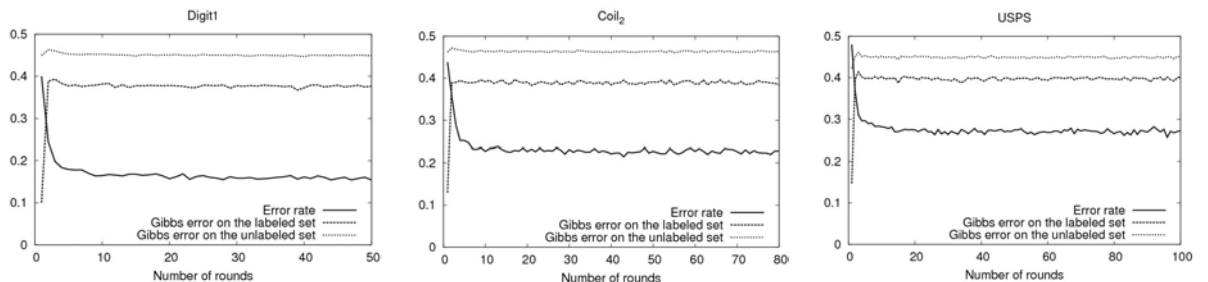

Figure 2: Classification error, train and test Gibbs errors with respect to the iterations of the SLA* algorithm for a fixed number of labeled training data $l = 10$.

In SLA$^*$ the automatic choice of the margin-threshold has the effect to select, at the first rounds of the algorithm, many unlabeled examples for which their class labels can be predicted with high confidence by the voted classifier. The exponential fall of the classification rate in figure 2 can be explained by the addition of these highly informative *pseudo-labeled* examples at the first steps of the learning process (figure 1). After this fall, few examples are added because the learning algorithm does not increase the margin on unlabeled data. Hence, the number of additional pseudo-labeled examples decreases resulting in a plateau in the classification error curves in figure 2. We further notice that the error of the Gibbs classifier on labeled data increases fastly to a stationary error point and that on the unlabeled examples does not vary in time.

## 5 Conclusions

The contribution of this paper is two fold. First, we proposed a bound on the risk of the voted classifier using the margin distribution of unlabeled examples and an estimation of the risk of the Gibbs classifier. We have shown that our bound is a good approximation of the true risk when the errors of the associated Gibbs classifier can accurately be estimated and that the voted classifier makes most its errors on low margin examples.

The proof of the bound passed through a second bound on the joint probability that the voted classifier makes an error and that the margin is above a given threshold. This tool led to the conditional probability that the voted classifier makes an error knowing that the margin is above a given threshold. We showed that the search of a margin threshold minimizing this conditional probability can be obtained by analytically solving a linear program.

This resolution conducted to our second contribution which is to find automatically the margin threshold in a self-learning algorithm. Empirical results on a number of data sets have shown that the adaptive threshold allows to enhance the performance of a self-learning algorithm.

## Footnotes

[1] The proofs can be inferred to the more general *noisy* case, but one has to replace the summation $\sum_{\mathbf{x}' \in X_{\mathcal{U}}}$ by $\sum_{(\mathbf{x}', y') \in X_{\mathcal{U}} \times \{-1, +1\}}. P_{(\mathbf{x}', y') \sim \mathcal{D}}(y' | \mathbf{x}')$ in the definitions of equations (3) and (4).

[2]Bennett *et al.* [1] have shown that the minimization of (19) allows to reach a local minima of the margin loss function $\mathcal{L}_M(H, Z_\ell, Z_{\mathcal{U}}) = \frac{1}{l} \sum_{\mathbf{x} \in Z_\ell} e^{-yH(\mathbf{x})} + \frac{1}{|Z_{\mathcal{U}}|} \sum_{\mathbf{x}' \in Z_{\mathcal{U}}} e^{|H(\mathbf{x}')|}$.

[3]http://www.kyb.tuebingen.mpg.de/ssl-book/benchmarks.html

## References

[1] Bennett, K., Demiriz, A. & Maclin, R. (2002) Expoliting unlabeled data in ensemble methods. In *Proc. ACM Int. Conf. Knowledge Discovery and Data Mining*, 289-296.

[2] Blake, C., Keogh, E. & Merz, C.J. (1998) UCI repository of machine learning databases. University of California, Irvine. [on-line] http://www.ics.uci.edu/ mlearn/MLRepository.html

[3] Chapelle, O., Schölkopf, B. & Zien, A. (2006) *Semi-supervised learning.* MA: MIT Press.

[4] Derbeko, P., El-Yaniv, R. & Meir, R. (2004) Explicit learning curves for transduction and application to clustering and compression algorithms. *Journal of Artificial Intelligence Research* **22**:117-142.

[5] Dietterich, T.G. (2000) Ensemble Methods in Machine Learning. In *First International Workshop on Multiple Classifier Systems*, 1-15.

[6] Grandvalet, Y. & Bengio, Y. (2005) Semi-supervised learning by entropy minimization. In *Advances in Neural Information Processing Systems 17*, 529-536. Cambridge, MA: MIT Press.

[7] Joachims, T. (1999) Transductive Inference for Text Classification using Support Vector Machines. In *Proceedings of the $16^{th}$ International Conference on Machine Learning*, 200-209.

[8] Lehmann, E.L. (1975) *Nonparametric Statistical Methods Based on Ranks.* McGraw-Hill, New York.

[9] McAllester, D. (2003) Simplified PAC-Bayesian margin bounds. In *Proc. od the $16^{th}$ Annual Conference on Learning Theory, Lecture Notes in Artificial Intelligence*, 203-215.

[10] Seeger, M. (2002) Learning with labeled and unlabeled data. *Technical report, Institute for Adaptive and Neural Computation, University of Edinburgh.*

[11] Tür, G., Hakkani-Tür, D.Z. & Schapire, R.E. (2005) Combining active and semi-supervised learning for spoken language understanding. *Journal of Speech Communication* **45**(2):171-186.

[12] Vittaut, J.-N., Amini, M.-R. & Gallinari, P. (2002) Learning Classification with Both Labeled and Unlabeled Data. In *European Conference on Machine Learning*, 468-476.
